# Asymptotic Universality for Learning Curves of Support Vector Machines

M. Opper[1]             R. Urbanczik[2]

[1] Neural Computing Research Group
School of Engineering and Applied Science
Aston University, Birmingham B4 7ET, UK.
opperm@aston.ac.uk

[2]Institut Für Theoretische Physik,
Universität Würzburg Am Hubland, D-97074 Würzburg, Germany
robert@physik.uni-wuerzburg.de.

## Abstract

Using methods of Statistical Physics, we investigate the rôle of model complexity in learning with support vector machines (SVMs). We show the advantages of using SVMs with kernels of infinite complexity on noisy target rules, which, in contrast to common theoretical beliefs, are found to achieve optimal generalization error although the training error does not converge to the generalization error. Moreover, we find a universal asymptotics of the learning curves which only depend on the target rule but not on the SVM kernel.

## 1 Introduction

Powerful systems for data inference, like neural networks implement complex input-output relations by learning from example data. The price one has to pay for the flexibility of these models is the need to choose the proper model complexity for a given task, i.e. the system architecture which gives good generalization ability for novel data. This has become an important problem also for *support vector machines* [1]. The main advantage of SVMs is that the learning task is a convex optimization problem which can be reliably solved even when the example data require the fitting of a very complicated function. A common argument in computational learning theory suggests that it is dangerous to utilize the *full flexibility* of the SVM to learn the training data perfectly when these contain an amount of noise. By fitting more and more noisy data, the machine may implement a rapidly oscillating function rather than the smooth mapping which characterizes most practical learning tasks. Its prediction ability could be no better than random guessing in that case. Hence, modifications of SVM training [2] that allow for training errors were suggested to be necessary for realistic noisy scenarios. This has the drawback of introducing extra model parameters and spoils much of the original elegance of SVMs.

Surprisingly, the results of this paper show that the picture is rather different in the important case of high dimensional data spaces. Using methods of Statistical Physics, we show that asymptotically, the SVM achieves optimal generalization ability for noisy data already for zero training error. Moreover, the asymptotic rate of decay of the generalization error is *universal*, i.e. independent of the kernel that defines the SVM. These results have been published previously only in a physics journal [3].

As is well known, SVMs classify inputs $y$ using a nonlinear mapping into a feature vector $\Psi(y)$ which is an element of a Hilbert space. Based on a training set of $m$ inputs $x^\mu$ and their desired classifications $\tau^\mu$, SVMs construct the maximum margin hyperplane $\mathcal{P}$ in the feature space. $\mathcal{P}$ can be expressed as a linear combination of the feature vectors $\Psi(x^\mu)$, and to classify an input $y$, that is to decide on which side of $\mathcal{P}$ the image $\Psi(y)$ lies, one basically has to evaluate inner products $\Psi(x^\mu) \cdot \Psi(y)$. For carefully chosen mappings $\Psi$ and Hilbert spaces, inner products $\Psi(x) \cdot \Psi(y)$ can be evaluated efficiently using a kernel function $k(x, y) = \Psi(x) \cdot \Psi(y)$, without having to individually calculate the feature vectors $\Psi(x)$ and $\Psi(y)$. In this manner it becomes computationally feasible to use very high and even infinite dimensional feature vectors.

This raises the intriguing question whether the use of a very high dimensional feature space may typically be helpful. So far, recent results [4, 5] obtained by using methods of Statistical Mechanics (which are naturally well suited for analysing stochastic models in high dimensional spaces), have been largely negative in this respect. They suggest (as one might perhaps expect) that it is rather important to match the complexity of the kernel to the target rule. The analysis in [4] considers the case of $N$-dimensional inputs with binary components and assumes that the target rule giving the correct classification $\tau$ of an input $x$ is obtained as the sign of a function $t(x)$ which is polynomial in the input components and of degree $L$. The SVM uses a kernel which is a polynomial of the inner product $x \cdot y$ in input space of degree $K \geq L$, and the feature space dimension is thus $\mathcal{O}(N^K)$. In this scenario it is shown, under mild regularity condition on the kernel and for large $N$, that the SVM generalizes well when the number of training examples $m$ is on the order of $N^L$. So the scale of the learning curve is determined by the complexity of the target rule and not by the kernel. However, considering the rate with which the generalization error approaches zero one finds the optimal $N^L/m$ decay only when $K$ is equal to $L$ and the convergence is substantially slower when $K > L$. So it is important to match the complexity of the kernel to the target rule and using a large value of $K$ is only justified if $L$ is assumed large and if one can use on the order of $N^L$ examples for training.

In this paper we show that the situation is very different when one considers the arguably more realistic scenario of a target rule corrupted by noise. Now one can no longer use $K = L$ since no separating hyperplane $\mathcal{P}$ will exist when $m$ is sufficiently large compared to $N^L$. However when $K > L$, this plane will exist and we will show that it achieves optimal generalization performance in the limit that $N^L/m$ is small. Remarkably, the asymptotic rate of decay of the generalization error is independent of the kernel in this case and a general characterization of the asymptote in terms of properties of the target rule is possible. In a second step we show that under mild regularity conditions these findings also hold when $k(x, y)$ is an arbitrary function of $x \cdot y$ or when the kernel is a function of the Euclidean distance $|x - y|$. The latter type of kernels is widely used in practical applications of SVMs.

## 2 Learning with Noise: Polynomial Kernels

We begin by assuming a polynomial kernel $k(x, y) = f(x \cdot y)$ where $f(z) = \sum_{k=0}^{K} c_k z^k$ is of degree $K$. Denoting by $\rho$ a multi-index $\rho = (\rho_1, \ldots, \rho_N)$ with $\rho_i \in \mathbb{N}_0$, we set $x_\rho = \sqrt{|\rho|!} \prod_{i=1}^{N} \frac{x_i^{\rho_i}}{\sqrt{\rho_i!}}$ and the degree of $x_\rho$ is $|\rho| = \sum_{i=1}^{N} \rho_i$. The kernel can then be described by features $\Psi_\rho(x) = \sqrt{c_{|\rho|}} x_\rho$ since $k(x, y) = \sum_\rho \Psi_\rho(x) \Psi_\rho(y)$, where the summation runs over all multi-indices of degree up to $K$. To assure that the features are real, we assume that the coefficients $c_k$ in the kernel are nonnegative. A hyperplane in feature space is parameterized by a weight vector $w$ with components $w_\rho$, and if $0 < \tau^\mu w \cdot \Psi(x^\mu)$, a point $(x^\mu, \tau^\mu)$ of the training set lies on the correct side of the plane. To express that the plane $\mathcal{P}$ has maximal distance to the points of the training set, we choose an arbitrary positive stability parameter $\kappa$ and require that the weight vector $w^*$ of $\mathcal{P}$ minimize $w \cdot w$ subject to the constraints $\kappa < \tau^\mu w \cdot \Psi(x^\mu)$, for $\mu = 1, \ldots, m$.

### 2.1 The Statistical Mechanics Formulation

Statistical Mechanics allows to analyze a variety of learning scenarios *exactly* in the "thermodynamic limit" of high input dimensionality, when the data distributions are simple enough. In this approach, one computes a partition function which serves as a generating function for important averages such as the generalization error. To define the partition problem for SVMs one first analyzes a soft version of the optimization problem characterized by an inverse temperature $\beta$. One considers the partition function

$$Z = \int dw \, e^{-\frac{1}{2}\beta w \cdot w} \prod_{\mu=1}^{m} \Theta(\tau^\mu w \cdot \Psi(x^\mu) - \kappa), \qquad (1)$$

where the SVM constraints are enforced strictly using the Heaviside step function $\Theta$. Properties of $w^*$ can be computed from $\ln Z$ and taking the limit $\beta \to \infty$.

To model the training data, we assume that the random and independent input components have zero mean and variance $1/N$. This scaling assures that the variance of $w \cdot \Psi(x^\mu)$ stays finite in the large $N$ limit. For the target rule we assume that its deterministic part is given by the polynomial $t(x) = \sum_\rho \sqrt{c_{|\rho|}} B_\rho x_\rho$ with real parameters $B_\rho$. The magnitude of the contribution of each degree $k$ to the value of $t(x)$ is measured by the quantities

$$T_k = c_k \frac{1}{N_k} \sum_{\rho, |\rho|=k} B_\rho^2 \qquad (2)$$

where $N_k = \binom{N+k-1}{k}$ is the number of terms in the sum. The degree of $t(x)$ is $L$ and lower than $K$, so $T_L > 0$ and $T_{L+1} = \ldots = T_K = 0$. Note, that this definition of $t(x)$ ensures that any feature necessary for computing $t(x)$ is available to the SVM. For brevity we assume that the constant term in $t(x)$ vanishes ($T_0 = 0$) and the normalization is $\sum_k T_k = 1$.

### 2.2 The Noise Model

In the deterministic case the label of a point $x$ would simply be the sign of $t(x)$. Here we consider a nondeterministic rule and the output label is obtained using a random variable $\tau_u \in \{-1, 1\}$ parameterized by a scalar $u$. The observable instances of the rule, and in particular the elements of the training set, are then obtained by

independently sampling the random variable $(x, \tau_{t(x)})$. Simple examples are additive noise, $\tau_{t(x)} = \text{sgn}(t(x) + \eta)$, or multiplicative noise, $\tau_{t(x)} = \text{sgn}(t(x)\eta)$, where $\eta$ is a noise term independent of $x$. In general, we will assume that the noise does not systematically corrupt the deterministic component, formally

$$1 > \text{Prob}\left(\tau_u = \text{sgn}(u)\right) > \frac{1}{2} \quad \text{for all } u. \tag{3}$$

So $\text{sgn}(t(x))$ is the best possible prediction of the output label of $x$, and the minimal achievable generalization error is $\epsilon_{\min} = \langle \Theta(-t(x)\tau_{t(x)}) \rangle_x$. In the limit of many input dimensions $N$, a central limit argument yields that for a typical target rule $\epsilon_{\min} = 2\langle \Theta(-u)\hat{\Theta}(u) \rangle_u$, where u is zero mean and unit variance Gaussian. The function $\hat{\Theta}$ will play a considerable rôle in the sequel. It is a symmetrized form of the probability $p(u)$ that $\tau_u$ is equal to 1, $\hat{\Theta}(u) = \frac{1}{2}(p(u) + 1 - p(-u))$.

## 2.3   Order Parameter Equations

One now evaluates the average of $\ln Z$ (Eq. 1) over random drawings of training data for large $N$ in terms of the order parameters

$$
\begin{aligned}
Q &= \left\langle \left\langle (w \cdot \Psi(x))^2 \right\rangle_x \right\rangle_w, \quad q = \left\langle (\langle w \rangle_w \cdot \Psi(x))^2 \right\rangle_x \quad \text{and} \\
r &= Q^{-\frac{1}{2}} \left\langle \langle w \cdot \Psi(x) \rangle_w B \cdot \Psi(x) \right\rangle_x.
\end{aligned}
\tag{4}
$$

Here the thermal average over $w$ refers to the Gibbs distribution (1). For the large $N$ limit, a scaling of the training set size $m$ must be specified, for which we make the generic Ansatz $m = \alpha N_l$, where $l = 1, \ldots, L$. Focusing on the limit of large $\beta$, where the density on the weight vectors converges to a delta peak and $q \to Q$, we introduce the rescaled order parameter $\chi = \beta(Q - q)/S_l$, with

$$S_l = f(1) - \sum_{i=0}^{l} c_i. \tag{5}$$

Note that this scaling with $S_l$ is only possible since the degree $K$ of the kernel $f(x \cdot y)$ is greater than $l$, and thus $S_l \neq 0$. Finally, we obtain an expression for $f_l = \lim_{\beta \to \infty} \lim_{N \to \infty} \langle\langle \ln Z \rangle\rangle S_l/(\beta N_l)$, where the double brackets denote averaging over all training sets of size $m$. The value of $f_l$ results from extremizing, with respect to $r, q$ and $\chi$, the function

$$
\begin{aligned}
f_l(r, q, \chi) = \\
\frac{-\alpha q}{\chi} \left\langle \hat{\Theta}(-u) G \left( ru + \sqrt{1-r^2}v - \frac{\kappa}{\sqrt{q}} \right) \right\rangle_{u,v} - \\
\frac{q}{2} \left( \frac{S_l}{c_l} - \frac{1}{\chi-1} \right) \left( 1 - \frac{r^2}{-(\chi-1)T_l S_l/c_l + \sum_{i=1}^{l} T_i} \right)
\end{aligned}
\tag{6}
$$

where $G(z) = \Theta(z)z^2$, and $u, v$ are independent Gaussian random variables with zero mean and unit variance.

Since the stationary value of $f_l$ is finite, $\langle\langle w^* \cdot w^* \rangle\rangle$ is of the order $N_l$. So the higher order components of $w^*$ are small, $(w_\rho^*)^2 \ll 1$ for $|\rho| > l$, although these components play a crucial rôle in ensuring that a hyperplane separating the training points exists even for large $\alpha$. But the key quantity obtained from Eq. (6) is the stationary value of $r$ which determines the generalization error of the SVM via $\epsilon_g = \langle \hat{\Theta}(-u)\Theta(ru + \sqrt{1-r^2}v) \rangle_{u,v}$, and in particular $\epsilon_g = \epsilon_{\min}$ for $r = 1$.

## 2.4 Universal Asymptotics

We now specialize to the case that $l$ equals $L$, the degree of the polynomial $t(x)$ in the target rule. So $m = \alpha N_L$ and for large $\alpha$, after some algebra, Eq. (6) yields

$$r = 1 - \frac{A(q^*)}{4B(q^*)^2} \frac{1}{\alpha} \tag{7}$$

where $B(q) = \left\langle \hat{\theta}(y)\theta\left(-y + \kappa/\sqrt{q}\right)\right\rangle_y$ and $A(q) = \left\langle \hat{\theta}(y)\theta\left(-y + \kappa/\sqrt{q}\right)\left(-y + \kappa/\sqrt{q}\right)^2\right\rangle_y$. Further $q^* = \arg\min_q qA(q)$, and considering the derivatives of $qA(q)$ for $q \to 0$ and $q \to \infty$, one may show that condition (3) assures that $qA(q)$ does have a minimum.

Equation (7) shows that optimal generalization performance is achieved on this scale in the limit of large $\alpha$. Remarkably, as long as $K > L$, the asymptote is invariant to the choice of the kernel since $A(q)$ and $B(q)$ are defined solely in terms of the target rule.

## 3 Extension to other Kernels

Our next goal is to understand cases where the kernel is a general function of the inner product or of the distance between the vectors. We still assume that the target rule is of finite complexity, i.e. defined by a polynomial and corrupted by noise and that the number of examples is polynomial in $N$. Remarkably, the more general kernels then reduce to the polynomial case in the thermodynamic limit.

Since it is difficult to find a description of the Hilbert space for $k(x, y)$ which is useful for a Statistical Physics calculation, our starting point is the dual representation: The weight vector $w^*$ defining the maximal margin hyperplane $\mathcal{P}$ can be written as a linear combination of the feature vectors $\Psi(x^\mu)$ and hence $w^* \cdot \Psi(y) = \sigma(y)$, where

$$\sigma(y) = \sum_{\mu=1}^{m} \lambda_\mu \tau^\mu k(x^\mu, y). \tag{8}$$

By standard results of convex optimization theory the $\lambda_\mu$ are uniquely defined by the Kuhn-Tucker conditions $\lambda_\mu \geq 0$, $\tau^\mu \sigma(x^\mu) \geq \kappa$ (for all patterns), further requiring that for positive $\lambda_\mu$ the second of the two inequalities holds as an equality. One also finds that $w^* \cdot w^* = \sum_{\mu=1}^{m} \lambda_\mu$ and for a polynomial kernel we thus obtain a bound on $\sum_{\mu=1}^{m} \lambda_\mu$ since $w^* \cdot w^*$ is $\mathcal{O}(m)$.

We first consider kernels $\phi(x \cdot y)$, with a general continuous function $\phi$ of the inner product, and assume that $\phi$ can be approximated by a polynomial $f$ in the sense that $\phi(1) = f(1)$ and $\phi(z) - f(z) = \mathcal{O}(z^K)$ for $z \to 0$. Now, with a probability approaching 1 with increasing $N$, the magnitude of $x^\mu \cdot x^\nu$ is smaller than, say, $N^{-1/3}$ for all different indices $\mu$ and $\nu$ as long as $m$ is polynomial in $N$. So, considering Eq. (8), for large $N$ the functions $\phi(z)$ and $f(z)$ will only be evaluated in a small region around zero and at $z = 1$ when used as kernels of a SVM trained on $m = \alpha N_L$ examples. Using the fact that $\sum_{\mu=1}^{m} \lambda_\mu = \mathcal{O}(m)$ we conclude that for large $N$ and $K > 3L$ the solution of the Kuhn-Tucker conditions for $f$ converges to the one for $\phi$. So Eqs. (6,7) can be used to calculate the generalization error for $\phi$ by setting $\mu_l = \phi^{(l)}(0)/l!$ for $l = 1, \ldots, L$, when $\phi$ is an analytic function. Note that results in [4] assure that $\mu_l \geq 0$ if the kernel $\phi(x \cdot y)$ is positive definite for all input dimensions $N$. Further, the same reduction to the polynomial case will hold in many instances where $\phi$ is not analytical but just sufficiently smooth close to 0.

### 3.1  RBF Kernels

We next turn to radial basis function (RBF) kernels where $k(x, y)$ depends only on the Euclidean distance between two inputs, $k(x, y) = \Phi(|x - y|^2)$. For binary input components ($x_i = \pm N^{-1/2}$) this is just the inner product case since $\Phi(|x - y|^2) = \Phi(2 - 2x \cdot y)$. However, for more general input distributions, e.g. Gaussian input components, the fluctuations of $|x|$ around its mean value 1 have the same magnitude as $x \cdot y$ even for large $N$, and an equivalence with inner product kernels is not evident.

Our starting point is the observation that any kernel $\Phi(|x - y|^2)$ which is positive definite for all input dimensions $N$ is a positive mixture of Gaussians [6]. More precisely $\Phi(z) = \int_0^\infty e^{-k^2 z} \, da(k)$ where the transform $a(k)$ is nondecreasing. For the special case of a single Gaussian one easily obtains features $\Psi_\rho$ by rewriting $\Phi(|x - y|^2) = e^{-|x-y|^2/2} = e^{-|x|^2/2} e^{x \cdot y} e^{-|y|^2/2}$. Expanding the kernel $e^{x \cdot y}$ into polynomial features, yields the features $\Psi_\rho(x) = e^{-|x|^2/2} x_\rho / \sqrt{|\rho|!}$ for $\Phi(|x - y|^2)$. But, for a generic weight vector $w$ in feature space,

$$w \cdot \Psi(x) = \sum_\rho w_\rho \Psi_\rho(x) = e^{-\frac{1}{2}|x|^2} \sum_\rho w_\rho \frac{x_\rho}{\sqrt{|\rho|!}} \qquad (9)$$

is of order 1, and thus for large $N$ the fluctuations of $|x|$ can be neglected.

This line of argument can be extended to the case that the kernel is a finite mixture of Gaussians, $\Phi(z) = \sum_{i=1}^n a_i e^{-\gamma_i^2 z/2}$ with positive coefficients $a_i$. Applying the reasoning for a single Gaussian to each term in the sum, one obtains a doubly indexed feature vector with components $\Psi_{\rho,i}(x) = e^{-\gamma_i^2 |x|^2/2} (a_i \gamma_i^{2|\rho|} / |\rho|!)^{1/2} x_\rho$. It is then straightforward to adapt the calculation of the partition function (Eq. 1 - 6) to the doubly indexed features, showing that the kernel $\Phi(|x - y|^2)$ yields the same generalization behavior as the inner product kernel $\Phi(2 - 2x \cdot y)$. Based on the calculation, we expect the same equivalence to hold for general radial basis function kernels, i.e. an infinite mixture of Gaussians, even if it would be involved to prove that the limit of many Gaussians commutes with the large $N$ limit.

## 4  Simulations

To illustrate the general results we first consider a scenario where a linear target rule, corrupted by additive Gaussian noise, is learned using different transcendental RBF kernels (Fig. 1). While Eq. (7) predicts that the asymptote of the generalization error does not depend on the kernel, remarkably, the dependence on the kernel is very weak for all values of $\alpha$. In contrast, the generalization error depends substantially on the nature of the noise process. Figure 2 shows the finding for a quadratic rule with additive noise for the cases that the noise is Gaussian and binary. In the Gaussian case a $1/\alpha$ decay of $\epsilon_g$ to $\epsilon_{\min}$ is found, whereas for binary noise the decay is exponential in $\alpha$. Note that in both cases the order parameter $r$ approaches 1 as $1/\alpha$.

## 5  Summary

The general characterization of learning curves obtained in this paper demonstrates that support vector machines with high order or even transcendental kernels have definitive advantages when the training data is noisy. Further the calculations leading to Eq. (6) show that maximizing the margin is an essential ingredient of the

approach: If one just chooses any hyperplane which classifies the training data correctly, the scale of the learning curve is not determined by the target rule and far more examples are needed to achieve good generalization. Nevertheless our findings are at odds with many of the current theoretical motivations for maximizing the margin which argue that this minimizes the effective Vapnik Chervonenkis dimension of the classifier and thus ensures fast convergence of the training error to the generalization error [1, 2]. Since we have considered hard margins, in contrast to the generalization error, the training error is zero, and we find no convergence between the two quantities. But close to optimal generalization is achieved since maximizing the margin biases the SVM to explain as much as possible of the data in terms of a low order polynomial. While the Statistical Physics approach used in this paper is only exactly valid in the thermodynamic limit, the numerical simulations indicate that the theory is already a good approximation for a realistic number of input dimensions.

We thank Rainer Dietrich for useful discussion and for giving us his code for the simulations. The work of M.O. was supported by the EPSRC (grant no. GR/M81601) and the British Council (ARC project 1037); R.U. was supported by the DFG and the DAAD.

# References

[1] C. Cortes and V. Vapnik. , *Machine Learning*, 20:273-297, 1995.

[2] N. Cristianini and J. Shawe-Taylor. *Support Vector Machines*. Cambridge University Press, 2000.

[3] M. Opper and R. Urbanczik. *Phys. Rev. Lett.*, 86:4410–4413, 2001.

[4] R. Dietrich, M. Opper, and H. Sompolinsky. *Phys. Rev. Lett.*, 82:2975 –2978, 1999.

[5] S. Risau-Gusman and M. Gordon. *Phys. Rev. E*, 62:7092–7099, 2000.

[6] I. Schoenberg. *Anal. Math*, 39:811–841, 1938.

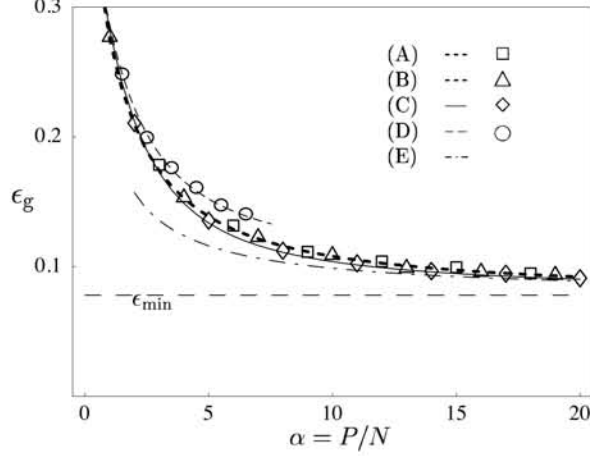

Figure 1: Linear target rule corrupted by additive Gaussian noise $\eta$ ($\langle\eta\rangle = 0$, $\langle\eta^2\rangle = 1/16$) and learned using different kernels. The curves are the theoretical prediction; symbols show simulation results for $N = 600$ with Gaussian inputs and error bars are approximately the size of the symbols. **(A)** Gaussian kernel, $\Phi(z) = e^{-kz}$ with k = 2/3. **(B)** Wiener kernel given by the nonanalytic function $\Phi(z) = e^{-c\sqrt{z}}$. We chose $c \approx 0.065$ so that the theoretical prediction for this case coincides with (A). **(C)** Gaussian kernel with $k = 1/20$, the influence of the parameter change on the learning curve is minimal. **(D)** Perceptron, $\phi(z) = z$. Above $\alpha_c \approx 7.5$ vanishing training error cannot be achieved and the SVM is undefined. **(E)** Kernel invariant asymptote for (A,B,C).

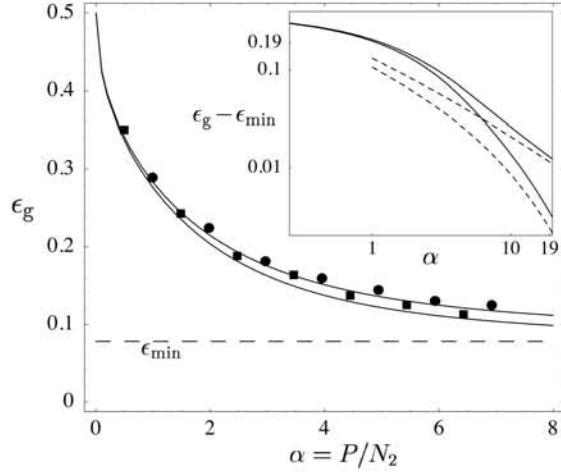

Figure 2: A noisy quadratic rule ($T_1 = 0$, $T_2 = 1$) learned using the Gaussian kernel with $k = 1/20$. The upper curve (simulations ●) is for additive Gaussian noise as in Fig. 1. The lower curve (simulations ■) is for binary noise, $\eta \pm s$ with equal probability. We chose $s \approx 0.20$ so that the value of $\epsilon_{min}$ is the same for the two noise processes. The inset shows that $\epsilon_g$ decays as $1/\alpha$ for Gaussian noise, whereas an exponential decay is found in the binary case. The dashed curves are the kernel invariant asymptotes. The simulations are for $N = 90$ with Gaussian inputs, and standard errors are approximately the size of the symbols.